# Regret Minimization in Games with Incomplete Information

**Martin Zinkevich**
maz@cs.ualberta.ca

**Michael Johanson**
johanson@cs.ualberta.ca

**Michael Bowling**
Computing Science Department
University of Alberta
Edmonton, AB Canada T6G2E8
bowling@cs.ualberta.ca

**Carmelo Piccione**
Computing Science Department
University of Alberta
Edmonton, AB Canada T6G2E8
carm@cs.ualberta.ca

## Abstract

Extensive games are a powerful model of multiagent decision-making scenarios with incomplete information. Finding a Nash equilibrium for very large instances of these games has received a great deal of recent attention. In this paper, we describe a new technique for solving large games based on regret minimization. In particular, we introduce the notion of counterfactual regret, which exploits the degree of incomplete information in an extensive game. We show how minimizing counterfactual regret minimizes overall regret, and therefore in self-play can be used to compute a Nash equilibrium. We demonstrate this technique in the domain of poker, showing we can solve abstractions of limit Texas Hold'em with as many as $10^{12}$ states, two orders of magnitude larger than previous methods.

## 1 Introduction

Extensive games are a natural model for sequential decision-making in the presence of other decision-makers, particularly in situations of imperfect information, where the decision-makers have differing information about the state of the game. As with other models (e.g., MDPs and POMDPs), its usefulness depends on the ability of solution techniques to scale well in the size of the model. Solution techniques for very large extensive games have received considerable attention recently, with poker becoming a common measuring stick for performance. Poker games can be modeled very naturally as an extensive game, with even small variants, such as two-player, limit Texas Hold'em, being impractically large with just under $10^{18}$ game states.

State of the art in solving extensive games has traditionally made use of linear programming using a realization plan representation [1]. The representation is linear in the number of game states, rather than exponential, but considerable additional technology is still needed to handle games the size of poker. Abstraction, both hand-chosen [2] and automated [3], is commonly employed to reduce the game from $10^{18}$ to a tractable number of game states (e.g., $10^7$), while still producing strong poker programs. In addition, dividing the game into multiple subgames each solved independently or in real-time has also been explored [2, 4]. Solving larger abstractions yields better approximate Nash equilibria in the original game, making techniques for solving larger games the focus of research in this area. Recent iterative techniques have been proposed as an alternative to the traditional linear programming methods. These techniques have been shown capable of finding approximate solutions to abstractions with as many as $10^{10}$ game states [5, 6, 7], resulting in the first significant improvement in poker programs in the past four years.

In this paper we describe a new technique for finding approximate solutions to large extensive games. The technique is based on regret minimization, using a new concept called counterfactual regret. We show that minimizing counterfactual regret minimizes overall regret, and therefore can be used to compute a Nash equilibrium. We then present an algorithm for minimizing counterfactual regret in poker. We use the algorithm to solve poker abstractions with as many as $10^{12}$ game states, two orders of magnitude larger than previous methods. We also show that this translates directly into an improvement in the strength of the resulting poker playing programs. We begin with a formal description of extensive games followed by an overview of regret minimization and its connections to Nash equilibria.

## 2 Extensive Games, Nash Equilibria, and Regret

Extensive games provide a general yet compact model of multiagent interaction, which explicitly represents the often sequential nature of these interactions. Before presenting the formal definition, we first give some intuitions. The core of an extensive game is a game tree just as in perfect information games (e.g., Chess or Go). Each non-terminal game state has an associated player choosing actions and every terminal state has associated payoffs for each of the players. The key difference is the additional constraint of information sets, which are sets of game states that the controlling player cannot distinguish and so must choose actions for all such states with the same distribution. In poker, for example, the first player to act does not know which cards the other players were dealt, and so all game states immediately following the deal where the first player holds the same cards would be in the same information set. We now describe the formal model as well as notation that will be useful later.

**Definition 1** *[8, p. 200] a finite extensive game with imperfect information has the following components:*

- *A finite set $N$ of **players**.*

- *A finite set $H$ of sequences, the possible **histories** of actions, such that the empty sequence is in $H$ and every prefix of a sequence in $H$ is also in $H$. $Z \subseteq H$ are the terminal histories (those which are not a prefix of any other sequences). $A(h) = \{a : (h, a) \in H\}$ are the actions available after a nonterminal history $h \in H$,*

- *A function $P$ that assigns to each nonterminal history (each member of $H \backslash Z$) a member of $N \cup \{c\}$. $P$ is the **player function**. $P(h)$ is the player who takes an action after the history $h$. If $P(h) = c$ then chance determines the action taken after history $h$.*

- *A function $f_c$ that associates with every history $h$ for which $P(h) = c$ a probability measure $f_c(\cdot|h)$ on $A(h)$ ($f_c(a|h)$ is the probability that $a$ occurs given $h$), where each such probability measure is independent of every other such measure.*

- *For each player $i \in N$ a partition $\mathcal{I}_i$ of $\{h \in H : P(h) = i\}$ with the property that $A(h) = A(h')$ whenever $h$ and $h'$ are in the same member of the partition. For $I_i \in \mathcal{I}_i$ we denote by $A(I_i)$ the set $A(h)$ and by $P(I_i)$ the player $P(h)$ for any $h \in I_i$. $\mathcal{I}_i$ is the **information partition** of player $i$; a set $I_i \in \mathcal{I}_i$ is an **information set** of player $i$.*

- *For each player $i \in N$ a utility function $u_i$ from the terminal states $Z$ to the reals $\mathbf{R}$. If $N = \{1, 2\}$ and $u_1 = -u_2$, it is a **zero-sum extensive game**. Define $\Delta_{u,i} = \max_z u_i(z) - \min_z u_i(z)$ to be the range of utilities to player $i$.*

Note that the partitions of information as described can result in some odd and unrealistic situations where a player is forced to forget her own past decisions. If all players can recall their previous actions and the corresponding information sets, the game is said to be one of **perfect recall**. This work will focus on finite, zero-sum extensive games with perfect recall.

### 2.1 Strategies

A **strategy of player $i$** $\sigma_i$ in an extensive game is a function that assigns a distribution over $A(I_i)$ to each $I_i \in \mathcal{I}_i$, and $\Sigma_i$ is the set of strategies for player $i$. A **strategy profile** $\sigma$ consists of a strategy for each player, $\sigma_1, \sigma_2, \ldots$, with $\sigma_{-i}$ referring to all the strategies in $\sigma$ except $\sigma_i$.

Let $\pi^\sigma(h)$ be the probability of history $h$ occurring if players choose actions according to $\sigma$. We can decompose $\pi^\sigma = \Pi_{i \in N \cup \{c\}} \pi_i^\sigma(h)$ into each player's contribution to this probability. Hence, $\pi_i^\sigma(h)$ is the probability that if player $i$ plays according to $\sigma$ then for all histories $h'$ that are a proper prefix of $h$ with $P(h') = i$, player $i$ takes the corresponding action in $h$. Let $\pi_{-i}^\sigma(h)$ be the product of all players' contribution (including chance) except player $i$. For $I \subseteq H$, define $\pi^\sigma(I) = \sum_{h \in I} \pi^\sigma(h)$, as the probability of reaching a particular information set given $\sigma$, with $\pi_i^\sigma(I)$ and $\pi_{-i}^\sigma(I)$ defined similarly.

The overall value to player $i$ of a strategy profile is then the expected payoff of the resulting terminal node, $u_i(\sigma) = \sum_{h \in Z} u_i(h) \pi^\sigma(h)$.

## 2.2 Nash Equilibrium

The traditional solution concept of a two-player extensive game is that of a Nash equilibrium. A **Nash equilibrium** is a strategy profile $\sigma$ where

$$u_1(\sigma) \geq \max_{\sigma_1' \in \Sigma_1} u_1(\sigma_1', \sigma_2) \qquad\qquad u_2(\sigma) \geq \max_{\sigma_2' \in \Sigma_2} u_2(\sigma_1, \sigma_2'). \tag{1}$$

An approximation of a Nash equilibrium or $\epsilon$-**Nash equilibrium** is a strategy profile $\sigma$ where

$$u_1(\sigma) + \epsilon \geq \max_{\sigma_1' \in \Sigma_1} u_1(\sigma_1', \sigma_2) \qquad\qquad u_2(\sigma) + \epsilon \geq \max_{\sigma_2' \in \Sigma_2} u_2(\sigma_1, \sigma_2'). \tag{2}$$

## 2.3 Regret Minimization

Regret is an online learning concept that has triggered a family of powerful learning algorithms. To define this concept, first consider repeatedly playing an extensive game. Let $\sigma_i^t$ be the strategy used by player $i$ on round $t$. The **average overall regret** of player $i$ at time $T$ is:

$$R_i^T = \frac{1}{T} \max_{\sigma_i^* \in \Sigma_i} \sum_{t=1}^{T} \left( u_i(\sigma_i^*, \sigma_{-i}^t) - u_i(\sigma^t) \right) \tag{3}$$

Moreover, define $\bar{\sigma}_i^t$ to be the average strategy for player $i$ from time 1 to $T$. In particular, for each information set $I \in \mathcal{I}_i$, for each $a \in A(I)$, define:

$$\bar{\sigma}_i^t(I)(a) = \frac{\sum_{t=1}^{T} \pi_i^{\sigma^t}(I) \sigma^t(I)(a)}{\sum_{t=1}^{T} \pi_i^{\sigma^t}(I)}. \tag{4}$$

There is a well-known connection between regret and the Nash equilibrium solution concept.

**Theorem 2** *In a zero-sum game at time $T$, if both player's average overall regret is less than $\epsilon$, then $\bar{\sigma}^T$ is a $2\epsilon$ equilibrium.*

An algorithm for selecting $\sigma_i^t$ for player $i$ is regret minimizing if player $i$'s average overall regret (regardless of the sequence $\sigma_{-i}^t$) goes to zero as $t$ goes to infinity. As a result, regret minimizing algorithms in self-play can be used as a technique for computing an approximate Nash equilibrium. Moreover, an algorithm's bounds on the average overall regret bounds the rate of convergence of the approximation.

Traditionally, regret minimization has focused on bandit problems more akin to normal-form games. Although it is conceptually possible to convert any finite extensive game to an equivalent normal-form game, the exponential increase in the size of the representation makes the use of regret algorithms on the resulting game impractical. Recently, Gordon has introduced the Lagrangian Hedging (LH) family of algorithms, which can be used to minimize regret in extensive games by working with the realization plan representation [5]. We also propose a regret minimization procedure that exploits the compactness of the extensive game. However, our technique doesn't require the costly quadratic programming optimization needed with LH allowing it to scale more easily, while achieving even tighter regret bounds.

# 3 Counterfactual Regret

The fundamental idea of our approach is to decompose overall regret into a set of additive regret terms, which can be minimized independently. In particular, we introduce a new regret concept for extensive games called counterfactual regret, which is defined on an individual information set. We show that overall regret is bounded by the sum of counterfactual regret, and also show how counterfactual regret can be minimized at each information set independently.

We begin by considering one particular information set $I \in \mathcal{I}_i$ and player $i$'s choices made in that information set. Define $u_i(\sigma, h)$ to be the expected utility given that the history $h$ is reached and then all players play using strategy $\sigma$. Define **counterfactual utility** $u_i(\sigma, I)$ to be the expected utility given that information set $I$ is reached and all players play using strategy $\sigma$ except that player $i$ plays to reach $I$, formally if $\pi^\sigma(h, h')$ is the probability of going from history $h$ to history $h'$, then:

$$u_i(\sigma, I) = \frac{\sum_{h \in I, h' \in Z} \pi^\sigma_{-i}(h) \pi^\sigma(h, h') u_i(h')}{\pi^\sigma_{-i}(I)} \tag{5}$$

Finally, for all $a \in A(I)$, define $\sigma|_{I \to a}$ to be a strategy profile identical to $\sigma$ except that player $i$ always chooses action $a$ when in information set $I$. The **immediate counterfactual regret** is:

$$R^T_{i,\text{imm}}(I) = \frac{1}{T} \max_{a \in A(I)} \sum_{t=1}^{T} \pi^{\sigma^t}_{-i}(I) \left( u_i(\sigma^t|_{I \to a}, I) - u_i(\sigma^t, I) \right) \tag{6}$$

Intuitively, this is the player's regret in its decisions at information set $I$ in terms of counterfactual utility, with an additional weighting term for the counterfactual probability that $I$ would be reached on that round *if the player had tried to do so.* As we will often be most concerned about regret when it is positive, let $R^{T,+}_{i,\text{imm}}(I) = \max(R^T_{i,\text{imm}}(I), 0)$ be the positive portion of immediate counterfactual regret.

We can now state our first key result.

**Theorem 3** $R^T_i \leq \sum_{I \in \mathcal{I}_i} R^{T,+}_{i,\text{imm}}(I)$

The proof is in the full version. Since minimizing immediate counterfactual regret minimizes the overall regret, it enables us to find an approximate Nash equilibrium if we can only minimize the immediate counterfactual regret.

The key feature of immediate counterfactual regret is that it can be minimized by controlling only $\sigma_i(I)$. To this end, we can use Blackwell's algorithm for approachability to minimize this regret independently on each information set. In particular, we maintain for all $I \in \mathcal{I}_i$, for all $a \in A(I)$:

$$R^T_i(I, a) = \frac{1}{T} \sum_{t=1}^{T} \pi^{\sigma^t}_{-i}(I) \left( u_i(\sigma^t|_{I \to a}, I) - u_i(\sigma^t, I) \right) \tag{7}$$

Define $R^{T,+}_i(I, a) = \max(R^T_i(I, a), 0)$, then the strategy for time $T + 1$ is:

$$\sigma^{T+1}_i(I)(a) = \begin{cases} \frac{R^{T,+}_i(I,a)}{\sum_{a \in A(I)} R^{T,+}_i(I,a)} & \text{if } \sum_{a \in A(I)} R^{T,+}_i(I, a) > 0 \\ \frac{1}{|A(I)|} & \text{otherwise.} \end{cases} \tag{8}$$

In other words, actions are selected in proportion to the amount of positive counterfactual regret for not playing that action. If no actions have any positive counterfactual regret, then the action is selected randomly. This leads us to our second key result.

**Theorem 4** *If player $i$ selects actions according to Equation 8 then $R^T_{i,\text{imm}}(I) \leq \Delta_{u,i} \sqrt{|A_i|}/\sqrt{T}$ and consequently $R^T_i \leq \Delta_{u,i} |\mathcal{I}_i| \sqrt{|A_i|}/\sqrt{T}$ where $|A_i| = \max_{h:P(h)=i} |A(h)|$.*

The proof is in the full version. This result establishes that the strategy in Equation 8 can be used in self-play to compute a Nash equilibrium. In addition, the bound on the average overall regret is linear in the number of information sets. These are similar bounds to what's achievable by Gordon's Lagrangian Hedging algorithms. Meanwhile, minimizing counterfactual regret does not require a costly quadratic program projection on each iteration. In the next section we demonstrate our technique in the domain of poker.

# 4 Application To Poker

We now describe how we use counterfactual regret minimization to compute a near equilibrium solution in the domain of poker. The poker variant we focus on is heads-up limit Texas Hold'em, as it is used in the AAAI Computer Poker Competition [9]. The game consists of two players (zero-sum), four rounds of cards being dealt, and four rounds of betting, and has just under $10^{18}$ game states [2]. As with all previous work on this domain, we will first abstract the game and find an equilibrium of the abstracted game. In the terminology of extensive games, we will merge information sets; in the terminology of poker, we will bucket card sequences. The quality of the resulting near equilibrium solution depends on the coarseness of the abstraction. In general, the less abstraction used, the higher the quality of the resulting strategy. Hence, the ability to solve a larger game means less abstraction is required, translating into a stronger poker playing program.

## 4.1 Abstraction

The goal of abstraction is to reduce the number of information sets for each player to a tractable size such that the abstract game can be solved. Early poker abstractions [2, 4] involved limiting the possible sequences of bets, e.g., only allowing three bets per round, or replacing all first-round decisions with a fixed policy. More recently, abstractions involving full four round games with the full four bets per round have proven to be a significant improvement [7, 6]. We also will keep the full game's betting structure and focus abstraction on the dealt cards.

Our abstraction groups together observed card sequences based on a metric called hand strength squared. Hand strength is the expected probability of winning[1] given only the cards a player has seen. This was used a great deal in previous work on abstraction [2, 4]. Hand strength squared is the expected square of the hand strength after the last card is revealed, given only the cards a player has seen. Intuitively, hand strength squared is similar to hand strength but gives a bonus to card sequences whose eventual hand strength has higher variance. Higher variance is preferred as it means the player eventually will be more certain about their ultimate chances of winning prior to a showdown. More importantly, we will show in Section 5 that this metric for abstraction results in stronger poker strategies.

The final abstraction is generated by partitioning card sequences based on the hand strength squared metric. First, all round-one card sequences (i.e., all private card holdings) are partitioned into ten equally sized buckets based upon the metric. Then, all round-two card sequences *that shared a round-one bucket* are partitioned into ten equally sized buckets based on the metric now applied at round two. Thus, a partition of card sequences in round two is a pair of numbers: its bucket in the previous round and its bucket in the current round given its bucket in the previous round. This is repeated after reach round, continuing to partition card sequences that agreed on the previous rounds' buckets into ten equally sized buckets based on the metric applied in that round. Thus, card sequences are partitioned into **bucket sequences**: a bucket from $\{1, \ldots 10\}$ for each round. The resulting abstract game has approximately $1.65 \times 10^{12}$ game states, and $5.73 \times 10^{7}$ information sets. In the full game of poker, there are approximately $9.17 \times 10^{17}$ game states and $3.19 \times 10^{14}$ information sets. So although this represents a significant abstraction on the original game it is two orders of magnitude larger than previously solved abstractions.

## 4.2 Minimizing Counterfactual Regret

Now that we have specified an abstraction, we can use counterfactual regret minimization to compute an approximate equilibrium for this game. The basic procedure involves having two players repeatedly play the game using the counterfactual regret minimizing strategy from Equation 8. After $T$ repetitions of the game, or simply *iterations*, we return $(\bar{\sigma}_1^T, \bar{\sigma}_2^T)$ as the resulting approximate equilibrium. Repeated play requires storing $R_i^t(I, a)$ for every information set $I$ and action $a$, and updating it after each iteration.[2]

For our experiments, we actually use a variation of this basic procedure, which exploits the fact that our abstraction has a small number of information sets relative to the number of game states. Although each information set is crucial, many consist of a hundred or more individual histories. This fact suggests it may be possible to get a good idea of the correct behavior for an information set by only sampling a fraction of the associated game states. In particular, for each iteration, we sample deterministic actions for the chance player. Thus, $\sigma_c^t$ is set to be a deterministic strategy, but chosen according to the distribution specified by $f_c$. For our abstraction this amounts to choosing a joint bucket sequence for the two players. Once the joint bucket sequence is specified, there are only 18,496 reachable states and 6,378 reachable information sets. Since $\pi_{-i}^{\sigma^t}(I)$ is zero for all other information sets, no updates need to be made for these information sets.[3]

This sampling variant allows approximately 750 iterations of the algorithm to be completed in a single second on a single core of a 2.4Ghz Dual Core AMD Opteron 280 processor. In addition, a straightforward parallelization is possible and was used when noted in the experiments. Since betting is public information, the flop-onward information sets for a particular preflop betting sequence can be computed independently. With four processors we were able to complete approximately 1700 iterations in one second. The complete algorithmic details with pseudocode can be found in the full version.

## 5    Experimental Results

Before discussing the results, it is useful to consider how one evaluates the strength of a near equilibrium poker strategy. One natural method is to measure the strategy's exploitability, or its performance against its worst-case opponent. In a symmetric, zero-sum game like heads-up poker[4], a perfect equilibrium has zero exploitability, while an $\epsilon$-Nash equilibrium has exploitability $\epsilon$. A convenient measure of exploitability is millibets-per-hand (mb/h), where a millibet is one thousandth of a small-bet, the fixed magnitude of bets used in the first two rounds of betting. To provide some intuition for these numbers, a player that always folds will lose 750 mb/h while a player that is 10 mb/h stronger than another would require over one million hands to be 95% certain to have won overall.

In general, it is intractable to compute a strategy's exploitability within the full game. For strategies in a reasonably sized abstraction it is possible to compute their exploitability within their own abstract game. Such a measure is a useful evaluation of the equilibrium computation technique that was used to generate the strategy. However, it does not imply the technique cannot be exploited by a strategy outside of its abstraction. It is therefore common to compare the performance of the strategy in the full game against a battery of known strong poker playing programs. Although positive expected value against an opponent is not transitive, winning against a large and diverse range of opponents suggests a strong program.

We used the sampled counterfactual regret minimization procedure to find an approximate equilibrium for our abstract game as described in the previous section. The algorithm was run for 2 billion iterations ($T = 2 \times 10^9$), or less than 14 days of computation when parallelized across four CPUs. The resulting strategy's exploitability within its own abstract game is 2.2 mb/h. After only 200 million iterations, or less than 2 days of computation, the strategy was already exploitable by less than 13 mb/h. Notice that the algorithm visits only 18,496 game states per iteration. After 200 million iterations each game state has been visited less than 2.5 times on average, yet the algorithm has already computed a relatively accurate solution.

### 5.1    Scaling the Abstraction

In addition to finding an approximate equilibrium for our large abstraction, we also found approximate equilibria for a number of smaller abstractions. These abstractions used fewer buckets per round to partition the card sequences. In addition to ten buckets, we also solved eight, six, and five

| Abs | Size $(\times 10^9)$ | Iterations $(\times 10^6)$ | Time (h) | Exp (mb/h) |
|---|---|---|---|---|
| 5 | 6.45 | 100 | 33 | 3.4 |
| 6 | 27.7 | 200 | 75 | 3.1 |
| 8 | 276 | 750 | 261 | 2.7 |
| 10 | 1646 | 2000 | 326† | 2.2 |

†: parallel implementation with 4 CPUs

(a)

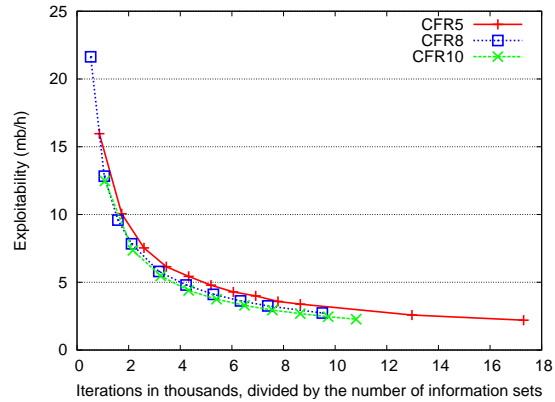

(b)

Figure 1: (a) Number of game states, number of iterations, computation time, and exploitability (in its own abstract game) of the resulting strategy for different sized abstractions. (b) Convergence rates for three different sized abstractions. The x-axis shows the number of iterations divided by the number of information sets in the abstraction.

bucket variants. As these abstractions are smaller, they require fewer iterations to compute a similarly accurate equilibrium. For example, the program computed with the five bucket approximation (CFR5) is about 250 times smaller with just under $10^{10}$ game states. After 100 million iterations, or 33 hours of computation without any parallelization, the final strategy is exploitable by 3.4 mb/h. This is approximately the same size of game solved by recent state-of-the-art algorithms [6, 7] with many days of computation.

Figure 1b shows a graph of the convergence rates for the five, eight, and ten partition abstractions. The y-axis is exploitability while the x-axis is the number of iterations normalized by the number of information sets in the particular abstraction being plotted. The rates of convergence almost exactly coincide showing that, in practice, the number of iterations needed is growing linearly with the number of information sets. Due to the use of sampled bucket sequences, the time per iteration is nearly independent of the size of the abstraction. This suggests that, in practice, the overall computational complexity is only linear in the size of the chosen card abstraction.

## 5.2  Performance in Full Texas Hold'em

We have noted that the ability to solve larger games means less abstraction is necessary, resulting in an overall stronger poker playing program. We have played our four near equilibrium bots with various abstraction sizes against each other and two other known strong programs: PsOpti4 and S2298. PsOpti4 is a variant of the equilibrium strategy described in [2]. It was the stronger half of Hyperborean, the AAAI 2006 Computer Poker Competition's winning program. It is available under the name SparBot in the entertainment program Poker Academy, published by BioTools. We have calculated strategies that exploit it at 175 mb/h. S2298 is the equilibrium strategy described in [6]. We have calculated strategies that exploit it at 52.5 mb/h. In terms of the size of the abstract game PsOpti4 is the smallest consisting of a small number of merged three round games. S2298 restricts the number of bets per round to 3 and uses a five bucket per round card abstraction based on hand-strength, resulting an abstraction slightly smaller than CFR5.

Table 1 shows a cross table with the results of these matches. Strategies from larger abstractions consistently, and significantly, outperform their smaller counterparts. The larger abstractions also consistently exploit weaker bots by a larger margin (e.g., CFR10 wins 19mb/h more from S2298 than CFR5).

Finally, we also played CFR8 against the four bots that competed in the bankroll portion of the 2006 AAAI Computer Poker Competition, which are available on the competition's benchmark server [9]. The results are shown in Table 2, along with S2298's previously published performance against the

|        | PsOpti4 | S2298 | CFR5 | CFR6 | CFR8 | CFR10 | Average |
|--------|---------|-------|------|------|------|-------|---------|
| PsOpti4 | 0 | -28 | -36 | -40 | -52 | -55 | -35 |
| S2298 | 28 | 0 | -17 | -24 | -30 | -36 | -13 |
| CFR5 | 36 | 17 | 0 | -5 | -13 | -20 | 2 |
| CFR6 | 40 | 24 | 5 | 0 | -9 | -14 | 7 |
| CFR8 | 52 | 30 | 13 | 9 | 0 | -6 | 16 |
| CFR10 | 55 | 36 | 20 | 14 | 6 | 0 | 22 |
| Max | 55 | 36 | 20 | 14 | 6 | 0 | |

Table 1: Winnings in mb/h for the row player in full Texas Hold'em. Matches with Opti4 used 10 duplicate matches of 10,000 hands each and are significant to 20 mb/h. Other matches used 10 duplicate matches of 500,000 hands each are are significant to 2 mb/h.

|        | Hyperborean | BluffBot | Monash | Teddy | Average |
|--------|-------------|----------|--------|-------|---------|
| S2298 | 61 | 113 | 695 | 474 | 336 |
| CFR8 | 106 | 170 | 746 | 517 | 385 |

Table 2: Winnings in mb/h for the row player in full Texas Hold'em.

same bots [6]. The program not only beats all of the bots from the competition but does so by a larger margin than S2298.

## 6   Conclusion

We introduced a new regret concept for extensive games called counterfactual regret. We showed that minimizing counterfactual regret minimizes overall regret and presented a general and poker-specific algorithm for efficiently minimizing counterfactual regret. We demonstrated the technique in the domain of poker, showing that the technique can compute an approximate equilibrium for abstractions with as many as $10^{12}$ states, two orders of magnitude larger than previous methods. We also showed that the resulting poker playing program outperforms other strong programs, including all of the competitors from the bankroll portion of the 2006 AAAI Computer Poker Competition.

## Footnotes

[1] Where a tie is considered "half a win"

[2] The bound from Theorem 4 for the basic procedure can actually be made significantly tighter in the specific case of poker. In the full version, we show that the bound for poker is actually independent of the size of the card abstraction.

[3]A regret analysis of this variant in poker is included in the full version. We show that the quadratic decrease in the cost per iteration only causes in a linear increase in the required number of iterations. The experimental results in the next section coincides with this analysis.

[4]A single hand of poker is not a symmetric game as the order of betting is strategically significant. However a pair of hands where the betting order is reversed is symmetric.

## References

[1] D. Koller and N. Megiddo. The complexity of two-person zero-sum games in extensive form. *Games and Economic Behavior*, pages 528–552, 1992.

[2] D. Billings, N. Burch, A. Davidson, R. Holte, J. Schaeffer, T. Schauenberg, and D. Szafron. Approximating game-theoretic optimal strategies for full-scale poker. In *International Joint Conference on Artificial Intelligence*, pages 661–668, 2003.

[3] A. Gilpin and T. Sandholm. Finding equilibria in large sequential games of imperfect information. In *ACM Conference on Electronic Commerce*, 2006.

[4] A. Gilpin and T. Sandholm. A competitive texas hold'em poker player via automated abstraction and real-time equilibrium computation. In *National Conference on Artificial Intelligence*, 2006.

[5] G. Gordon. No-regret algorithms for online convex programs. In *Neural Information Processing Systems 19*, 2007.

[6] M. Zinkevich, M. Bowling, and N. Burch. A new algorithm for generating strong strategies in massive zero-sum games. In *Proceedings of the Twenty-Seventh Conference on Artificial Intelligence (AAAI)*, 2007. To Appear.

[7] A. Gilpin, S. Hoda, J. Pena, and T. Sandholm. Gradient-based algorithms for finding nash equilibria in extensive form games. In *Proceedings of the Eighteenth International Conference on Game Theory*, 2007.

[8] M. Osborne and A. Rubenstein. *A Course in Game Theory*. The MIT Press, Cambridge, Massachusetts, 1994.

[9] M. Zinkevich and M. Littman. The AAAI computer poker competition. *Journal of the International Computer Games Association*, 29, 2006. News item.

